# How to Choose an Activation Function

**H. N. Mhaskar**
Department of Mathematics
California State University
Los Angeles, CA 90032
hmhaska@calstatela.edu

**C. A. Micchelli**
IBM Watson Research Center
P. O. Box 218
Yorktown Heights, NY 10598
cam@watson.ibm.com

## Abstract

We study the complexity problem in artificial feedforward neural networks designed to approximate real valued functions of several real variables; i.e., we estimate the number of neurons in a network required to ensure a given degree of approximation to every function in a given function class. We indicate how to construct networks with the indicated number of neurons evaluating standard activation functions. Our general theorem shows that the smoother the activation function, the better the rate of approximation.

## 1 INTRODUCTION

The approximation capabilities of feedforward neural networks with a single hidden layer has been studied by many authors, e.g., [1, 2, 5]. In [10], we have shown that such a network using practically any nonlinear activation function can approximate any continuous function of any number of real variables on any compact set to any desired degree of accuracy.

A central question in this theory is the following. If one needs to approximate a function from a known class of functions to a prescribed accuracy, how many neurons will be necessary to accomplish this approximation for all functions in the class? For example, Barron shows in [1] that it is possible to approximate any function satisfying certain conditions on its Fourier transform within an $L^2$ error of $\mathcal{O}(1/n)$ using a feedforward neural network with one hidden layer comprising of $n^2$ neurons, each with a sigmoidal activation function. On the contrary, if one is interested in a class of functions of $s$ variables with a bounded gradient on $[-1, 1]^s$,

then in order to accomplish this order of approximation, it is necessary to use at least $\mathcal{O}(n^s)$ number of neurons, *regardless of the activation function* (cf. [3]).

In this paper, our main interest is to consider the problem of approximating a function which is known only to have a certain number of smooth derivatives. We investigate the question of deciding which activation function will require how many neurons to achieve a given order of approximation for *all* such functions. We will describe a very general theorem and explain how to construct networks with various activation functions, such as the Gaussian and other radial basis functions advocated by Girosi and Poggio [13] as well as the classical squashing function and other sigmoidal functions.

In the next section, we develop some notation and briefly review some known facts about approximation order with a sigmoidal type activation function. In Section 3, we discuss our general theorem. This theorem is applied in Section 4 to yield the approximation bounds for various special functions which are commonly in use. In Section 5, we briefly describe certain dimension independent bounds similar to those due to Barron [1], but applicable with a general activation function. Section 6 summarizes our results.

## 2   SIGMOIDAL–TYPE ACTIVATION FUNCTIONS

In this section, we develop some notation and review certain known facts. For the sake of concreteness, we consider only uniform approximation, but our results are valid also for other $L^p$-norms with minor modifications, if any. Let $s \geq 1$ be the number of input variables. The class of all continuous functions on $[-1,1]^s$ will be denoted by $C^s$. The class of all $2\pi$- periodic continuous functions will be denoted by $C^{s*}$. The uniform norm in either case will be denoted by $\|\cdot\|$. Let $\Pi_{n,l,s,\sigma}$ denote the set of all possible outputs of feedforward neural networks consisting of $n$ neurons arranged in $l$ hidden layers and each neuron evaluating an activation function $\sigma$ where the inputs to the network are from $\mathbf{R}^s$. It is customary to assume more a priori knowledge about the target function than the fact that it belongs to $C^s$ or $C^{s*}$. For example, one may assume that it has continuous derivatives of order $r \geq 1$ and the sum of the norms of all the partial derivatives up to (and including) order $r$ is bounded. Since we are interested mainly in the relative error in approximation, we may assume that the target function is normalized so that this sum of the norms is bounded above by 1. The class of all the functions satisfying this condition will be denoted by $W_r^s$ (or $W_r^{s*}$ if the functions are periodic). In this paper, we are interested in the *universal approximation* of the classes $W_r^s$ (and their periodic versions). Specifically, we are interested in estimating the quantity

$$(2.1) \qquad\qquad \sup_{f \in W_r^s} E_{n,l,s,\sigma}(f)$$

where

$$(2.2) \qquad\qquad E_{n,l,s,\sigma}(f) := \inf_{P \in \Pi_{n,l,s,\sigma}} \|f - P\|.$$

The quantity $E_{n,l,s,\sigma}(f)$ measures the theoretically possible best order of approximation of an individual function $f$ by networks with $n$ neurons. We are interested

in determining the order that such a network can possibly achieve for *all* functions in the given class. An equivalent dual formulation is to estimate

(2.3) $$\tilde{E}_{n,l,s,\sigma}(W_r^s) := \min\{m \in \mathbf{Z} \ : \ \sup_{f \in W_r^s} E_{m,l,s,\sigma}(f) \leq 1/n\}.$$

This quantity measures the minimum number of neurons required to obtain accuracy of $1/n$ for all functions in the class $W_r^s$. An analogous definition is assumed for $W_r^{s*}$ in place of $W_r^s$.

Let $I\!H_n^s$ denote the class of all $s$-variable trigonometric polynomials of order at most $n$ and for a continuous function $f$, $2\pi$-periodic in each of its $s$ variables,

(2.4) $$E_n^s(f) := \min_{P \in I\!H_n^s} \|f - P\|.$$

We observe that $I\!H_n^s$ can be thought of as a subclass of all outputs of networks with a single hidden layer comprising of at most $(2n+1)^s$ neurons, each evaluating the activation function $\sin x$. It is then well known that

(2.5) $$\sup_{f \in W_r^{s*}} E_n^s(f) \leq cn^{-r}.$$

Here and in the sequel, $c, c_1, \cdots$ will denote positive constants independent of the functions and the number of neurons involved, but generally dependent on the other parameters of the problem such as $r$, $s$ and $\sigma$. Moreover, several constructions for the approximating trigonometric polynomials involved in (2.5) are also well known. In the dual formulation, (2.5) states that if $\sigma(x) := \sin x$ then

(2.6) $$\tilde{E}_{n,1,s,\sin}(W_r^{s*}) = \mathcal{O}(n^{s/r}).$$

It can be proved [3] that *any "reasonable" approximation process* that aims to approximate **all** functions in $W_r^{s*}$ up to an order of accuracy $1/n$ must necessarily depend upon at least $\mathcal{O}(n^{s/r})$ parameters. Thus, the activation function $\sin x$ provides optimal convergence rates for the class $W_r^{s*}$.

The problem of approximating an $r$ times continuously differentiable function $f : \mathbf{R}^s \to \mathbf{R}$ on $[-1,1]^s$ can be reduced to that of approximating another function from the corresponding periodic class as follows. We take an infinitely many times differentiable function $\psi$ which is equal to 1 on $[-2,2]^s$ and 0 outside of $[-\pi,\pi]^s$. The function $f\psi$ can then be extended as a $2\pi$-periodic function. This function is $r$ times continuously differentiable and its derivatives can be bounded by the derivatives of $f$ using the Leibnitz formula. A function that approximates this $2\pi$-periodic function also approximates $f$ on $[-1,1]^s$ with the same order of approximation. In contrast, it is not customary to choose the activation function to be periodic.

In [10] we introduced the notion of a *higher order* sigmoidal function as follows. Let $k \geq 0$. We say that a function $\sigma : \mathbf{R} \to \mathbf{R}$ is *sigmoidal of order $k$* if

(2.7) $$\lim_{x \to \infty} \frac{\sigma(x)}{x^k} = 1, \qquad \lim_{x \to -\infty} \frac{\sigma(x)}{x^k} = 0,$$

and

(2.8) $$|\sigma(x)| \leq c(1 + |x|)^k, \qquad x \in \mathbf{R}.$$

A sigmoidal function of order 0 is thus the customary bounded sigmoidal function.

We proved in [10] that for any integer $r \geq 1$ and a sigmoidal function $\sigma$ of order $r - 1$, we have

$$(2.9) \qquad \tilde{E}_{n,1,s,\sigma}(W_r^s) = \begin{cases} \mathcal{O}(n^{1/r}) & \text{if } s = 1, \\ \mathcal{O}(n^{s/r+(s+2r)/r^2}) & \text{if } s \geq 2. \end{cases}$$

Subsequently, Mhaskar showed in [6] that if $\sigma$ is a sigmoidal function of order $k \geq 2$ and $r \geq 1$ then, with $l = \mathcal{O}(\log r / \log k))$,

$$(2.10) \qquad \tilde{E}_{n,l,s,\sigma}(W_r^s) = \mathcal{O}(n^{s/r}).$$

Thus, an *optimal* network can be constructed using a sigmoidal function of higher order. During the course of the proofs in [10] and [6], we actually constructed the networks explicitly. The various features of these constructions from the connectionist point of view are discussed in [7, 8, 9].

In this paper, we take a different viewpoint. We wish to determine which activation function leads to what approximation order. As remarked above, for the approximation of periodic functions, the periodic activation function $\sin x$ provides an optimal network. Therefore, we will investigate the degree of approximation by neural networks first in terms of a general periodic activation function and then apply these results to the case when the activation function is not periodic.

## 3   A GENERAL THEOREM

In this section, we discuss the degree of approximation of periodic functions using periodic activation functions. It is our objective to include the case of radial basis functions as well as the usual "first order" neural networks in our discussion. To encompass both of these cases, we discuss the following general formuation. Let $s \geq d \geq 1$ be integers and $\phi \in C^{d*}$. We will consider the approximation of functions in $C^{s*}$ by linear combinations of quantities of the form $\phi(Ax + \mathbf{t})$ where $A$ is a $d \times s$ matrix and $\mathbf{t} \in \mathbf{R}^d$. (In general, both $A$ and $\mathbf{t}$ are parameters of the network.) When $d = s$, $A$ is the identity matrix and $\phi$ is a radial function, then a linear combination of $n$ such quantities represents the output of a radial basis function network with $n$ neurons. When $d = 1$ then we have the usual neural network with one hidden layer and periodic activation function $\phi$.

We define the Fourier coefficients of $\phi$ by the formula

$$(3.1) \qquad \hat{\phi}(\mathbf{m}) := \frac{1}{(2\pi)^d} \int_{[-\pi,\pi]^d} \phi(\mathbf{t}) e^{-i\mathbf{m}\cdot\mathbf{t}} d\mathbf{t}, \qquad \mathbf{m} \in \mathbf{Z}^d.$$

Let

$$(3.2) \qquad S_\phi \subseteq \{\mathbf{m} \in \mathbf{Z}^d \ : \ \hat{\phi}(\mathbf{m}) \neq 0\}$$

and *assume* that there is a set $J$ co ttaining $d \times s$ matrices with integer entries such that

$$(3.3) \qquad \mathbf{Z}^s = \{A^T \mathbf{m} \ : \ \mathbf{m} \in S_\phi, \ A \in J\}$$

where $A^T$ denotes the transpose of $A$. If $d = 1$ and $\hat{\phi}(1) \neq 0$ (*the neural network case*) then we may choose $S_\phi = \{1\}$ and $J$ to be $\mathbf{Z}^s$ (considered as row vectors). If $d = s$ and $\phi$ is a function with none of its Fourier coefficients equal to zero (*the radial basis case*) then we may choose $S_\phi = \mathbf{Z}^s$ and $J = \{I_{s\times s}\}$. For $\mathbf{m} \in \mathbf{Z}^s$, we let $\mathbf{k_m}$ be the multi-integer with minimum magnitude such that $\mathbf{m} = A^T\mathbf{k_m}$ for some $A = A_{\mathbf{m}} \in J$. Our estimates will need the quantities

$$(3.4) \qquad m_n := \min\{|\hat{\phi}(\mathbf{k_m})| \ : \ -2n \leq \mathbf{m} \leq 2n\}$$

and

$$(3.5) \qquad N_n := \max\{|\mathbf{k_m}| \ : \ -2n \leq \mathbf{m} \leq 2n\}$$

where $|\mathbf{k_m}|$ is the maximum absolute value of the components of $\mathbf{k_m}$. In the neural network case, we have $m_n = |\hat{\phi}(1)|$ and $N_n = 1$. In the radial basis case, $N_n = 2n$.

Our main theorem can be formulated as follows.

THEOREM 3.1. *Let $s \geq d \geq 1$, $n \geq 1$ and $N \geq N_n$ be integers, $f \in C^{s*}$, $\phi \in C^{d*}$. It is possible to construct a network*

$$(3.6) \qquad G_{n,N,\phi}(f;\mathbf{x}) := \sum d_{\mathbf{j}}\phi(A_{\mathbf{j}}\mathbf{x} + t_{\mathbf{j}})$$

*such that*

$$(3.7) \qquad \|f - G_{n,N,\phi}(f)\| \leq c\left\{E_n^s(f) + \frac{E_N^d(\phi)n^{s/2}}{m_n}\|f\|\right\}.$$

*In (3.6), the sum contains at most $\mathcal{O}(n^s N^d)$ terms, $A_{\mathbf{j}} \in J$, $t_{\mathbf{j}} \in \mathbf{R}^d$, and $d_{\mathbf{j}}$ are linear functionals of $f$, depending upon $n, N, \phi$.*

The estimate (3.7) relates the degree of approximation of $f$ by neural networks explicitly in terms of the degree of approximation of $f$ and $\phi$ by trigonometric polynomials. Well known estimates from approximation theory, such as (2.5), provide close connections between the smoothness of the functions involved and their degree of trigonometric polynomial approximation. In particular, (3.7) indicates that the smoother the function $\phi$ the better will be the degree of approximation.

In [11], we have given explicit constructions of the operator $G_{n,N,\phi}$. The formulas in [11] show that the network can be trained in a very simple manner, given the Fourier coefficients of the target function. The weights and thresholds (or the centers in the case of the radial basis networks) are determined universally for all functions being approximated. Only the coefficients at the output layer depend upon the function. Even these are given explicitly as linear combinations of the Fourier coefficients of the target function. The explicit formulas in [11] show that in the radial basis case, the operator $G_{n,N,\phi}$ actually contains only $\mathcal{O}(n + N)^s$ summands.

## 4  APPLICATIONS

In Section 3, we had assumed that the activation function $\phi$ is periodic. If the activation function $\sigma$ is not periodic, but satisfies certain decay conditions near

$\infty$, it is still possible to construct a periodic function for which Theorem 3.1 can be applied. Suppose that there exists a function $\psi$ in the linear span of $\mathcal{A}_{\sigma,J} := \{\sigma(Ax+t) : A \in J, \ t \in \mathbf{R}^d\}$, which is integrable on $\mathbf{R}^d$ and satisfies the condition that

$$(4.1) \qquad |\psi(\mathbf{x})| \le c\|\mathbf{x}\|^{-\tau}, \qquad \text{for some } \tau > d.$$

Under this assumption, the function

$$(4.2) \qquad \psi^\circ(\mathbf{x}) := \sum_{\mathbf{k}\in\mathbf{Z}^d} \psi(\mathbf{x} - 2\pi\mathbf{k})$$

is a $2\pi$-periodic function integrable on $[-\pi, \pi]^s$. We can then apply Theorem 3.1 with $\psi^\circ$ instead of $\phi$. In $G_{n,N,\psi^\circ}$, we next replace $\psi^\circ$ by a function obtained by judiciously truncating the infinite sum in (4.2). The error made in this replacement can be estimated using (4.1). Knowing the number of evaluations of $\sigma$ in the expression for $\psi$ as a finite linear combination of elements of $\mathcal{A}_{\sigma,J}$, we then have an estimate on the degree of approximation of $f$ in terms of the number of evaluations of $\sigma$. This process was applied on a number of functions $\sigma$. The results are summarized in Table 1.

## 5 DIMENSION INDEPENDENT BOUNDS

In this section, we describe certain estimates on the $L^2$ degree of approximation that are independent of the dimension of the input space. In this section, $\|\cdot\|$ denotes the $L^2$ norm on $[-1,1]^s$ (respectively $[-\pi,\pi]^s$) and we approximate functions in the class $SF_s$ defined by

$$(5.1) \qquad SF_s := \{f \in C^{s*} : \|f\|_{SF,s} := \sum_{\mathbf{m}\in\mathbf{Z}^s} |\hat{f}(\mathbf{m})| \le 1\}.$$

Analogous to the degree of approximation from $I\!H_n^s$, we define the $n$-th degree of approximation of a function $f \in C^{s*}$ by the formula

$$(5.2) \qquad \epsilon_{n,s}(f) := \inf_{\Lambda\subset\mathbf{Z}^s,|\Lambda|\le n} \|f - \sum_{\mathbf{m}\in\Lambda} \hat{f}(\mathbf{m})e^{i\mathbf{m}\cdot\mathbf{x}}\|$$

where we require the norm involved to be the $L^2$ norm. In (5.2), there is no need to assume that $n$ is an integer.

Let $\phi$ be a square integrable $2\pi$-periodic function of one variable. We define the $L^2$ degree of approximation by networks with a single hidden layer by the formula

$$(5.3) \qquad E^{(2)}_{\phi,n,s}(f) := \inf_{P\in\Pi_{m,1,s,\phi}} \|f - P\|$$

where $m$ is the largest integer not exceeding $n$. Our main theorem in this connection is the following

THEOREM 5.1. *Let $s \ge 1$ be an integer, $f \in SF_s$, $\phi \in L_1^2$ and $\hat{\phi}(1) \ne 0$. Then, for integers $n, N \ge 1$,*

$$(5.4) \qquad E^{(2)}_{\phi,2nN,s}(f) \le \{\frac{\delta_n}{\sqrt{n+1}} + \frac{2\epsilon_{N,1}(\phi)}{|\hat{\phi}(1)|}\}\|f\|_{SF,s}$$

Table 1: Order of magnitude of $\tilde{E}_{n,l,s,\sigma}(W_r^s)$ for different $\sigma$'s

| Function $\sigma$ | $\tilde{E}_{n,l,s,\sigma}$ | Remarks |
|---|---|---|
| Sigmoidal, order $r-1$ | $n^{1/r}$ | $s=d=1,\ l=1$ |
| Sigmoidal, order $r-1$ | $n^{s/r+(s+2r)/r^2}$ | $s\geq 2,\ d=1,\ l=1$ |
| $x^k$, if $x\geq 0$, 0, if $x<0$. | $n^{s/r+(2r+s)/2rk}$ | $k\geq 2,\ s\geq 2,\ d=1,\ l=1$ |
| $(1+e^{-x})^{-1}$ | $n^{s/r}(\log n)^2$ | $s\geq 2,\ d=1,\ l=1$ |
| Sigmoidal, order $k$ | $n^{s/r}$ | $k\geq 2,\ s\geq 1,\ d=1,$ $l=\mathcal{O}(\log r/\log k))$ |
| $\exp(-\|\mathbf{x}\|^2/2)$ | $n^{2s/r}$ | $s=d\geq 2, l=1$ |
| $\|\mathbf{x}\|^k(\log\|\mathbf{x}\|)^\delta$ | $n^{(s/r)(2+(3s+2r)/k)}$ | $s=d\geq 2,\ k>0,\ k+s$ even, $\delta=0$ if $s$ odd, 1 if $s$ even, $l=1$ |

*where $\{\delta_n\}$ is a sequence of positive numbers, $0\leq\delta_n\leq 2$, depending upon $f$ such that $\delta_n\to 0$ as $n\to\infty$. Moreover, the coefficients in the network that yields (5.4) are bounded, independent of $n$ and $N$.*

We may apply Theorem 5.1 in the same way as Theorem 3.1. For the squashing activation function, this gives an order of approximation $\mathcal{O}(n^{-1/2})$ with a network consisting of $n(\log n)^2$ neurons arranged in one hidden layer. With the truncated power function $x_+^k$ (cf. Table 1, entry 3) as the activation function, the same order of approximation is obtained with a network with a single hidden layer and $\mathcal{O}(n^{1+1/(2k)})$ neurons.

# 6  CONCLUSIONS.

We have obtained estimates on the number of neurons necessary for a network with a single hidden layer to provide a given accuracy of **all** functions under the only a priori assumption that the derivatives of the function up to a certain order should exist. We have proved a general theorem which enables us to estimate this number

in terms of the growth and smoothness of the activation function. We have explicitly constructed networks which provide the desired accuracy with the indicated number of neurons.

## Acknowledgements

The research of H. N. Mhaskar was supported in part by AFOSR grant 2-26 113.

## References

1. BARRON, A. R., *Universal approximation bounds for superposition of a sigmoidal function*, IEEE Trans. on Information Theory, **39**.

2. CYBENKO, G., *Approximation by superposition of sigmoidal functions*, Mathematics of Control, Signals and Systems, **2**, # 4 (1989), 303-314.

3. DeVORE, R., HOWARD, R. AND MICCHELLI, C.A., *Optimal nonlinear approximation*, Manuscripta Mathematica, **63** (1989), 469-478.

4. HECHT-NILESEN, R., *Thoery of the backpropogation neural network*, IEEE International Conference on Neural Networks, **1** (1988), 593-605.

5. HORNIK, K., STINCHCOMBE, M. AND WHITE, H., *Multilayer feedforward networks are universal approximators*, Neural Networks, **2** (1989), 359-366.

6. MHASKAR, H. N., *Approximation properties of a multilayered feedforward artificial neural network*, Advances in Computational Mathematics **1** (1993), 61-80.

7. MHASKAR, H. N., *Neural networks for localized approximation of real functions*, in "Neural Networks for Signal Processing, III", (Kamm, Huhn, Yoon, Chellappa and Kung Eds.), IEEE New York, 1993, pp. 190-196.

8. MHASKAR, H. N., *Approximation of real functions using neural networks*, in Proc. of Int. Conf. on Advances in Comput. Math., New Delhi, India, 1993, World Sci. Publ., H. P. Dikshit, C. A. Micchelli eds., 1994.

9. MHASKAR, H. N., *Noniterative training algorithms for neural networks*, Manuscript, 1993.

10. MHASKAR, H. N. AND MICCHELLI, C. A., *Approximation by superposition of a sigmoidal function and radial basis functions*, Advances in Applied Mathematics, **13** (1992), 350-373.

11. MHASKAR, H. N. AND MICCHELLI, C. A., *Degree of approximation by superpositions of a fixed function*, in preparation.

12. MHASKAR, H. N. AND MICCHELLI, C. A., *Dimension independent bounds on the degree of approximation by neural networks*, Manuscript, 1993.

13. POGGIO, T. AND GIROSI, F., *Regularization algorithms for learning that are equivalent to multilayer networks*, Science, **247** (1990), 978-982.
